# Preconditioner Approximations for Probabilistic Graphical Models

**Pradeep Ravikumar** **John Lafferty**
School of Computer Science
Carnegie Mellon University

## Abstract

We present a family of approximation techniques for probabilistic graphical models, based on the use of graphical preconditioners developed in the scientific computing literature. Our framework yields rigorous upper and lower bounds on event probabilities and the log partition function of undirected graphical models, using non-iterative procedures that have low time complexity. As in mean field approaches, the approximations are built upon tractable subgraphs; however, we recast the problem of optimizing the tractable distribution parameters and approximate inference in terms of the well-studied linear systems problem of obtaining a good matrix preconditioner. Experiments are presented that compare the new approximation schemes to variational methods.

## 1   Introduction

Approximate inference techniques are enabling sophisticated new probabilistic models to be developed and applied to a range of practical problems. One of the primary uses of approximate inference is to estimate the partition function and event probabilities for undirected graphical models, which are natural tools in many domains, from image processing to social network modeling. A central challenge is to improve the accuracy of existing approximation methods, and to derive rigorous rather than heuristic bounds on probabilities in such graphical models. In this paper, we present a simple new approach to the approximate inference problem, based upon non-iterative procedures that have low time complexity. We follow the variational mean field intuition of focusing on tractable subgraphs, however we recast the problem of optimizing the tractable distribution parameters as a generalized linear system problem. In this way, the task of deriving a tractable distribution conveniently reduces to the well-studied problem of obtaining a good *preconditioner* for a matrix (Boman and Hendrickson, 2003). This framework has the added advantage that tighter bounds can be obtained by reducing the sparsity of the preconditioners, at the expense of increasing the time complexity for computing the approximation.

In the following section we establish some notation and background. In Section 3, we outline the basic idea of our proposed framework, and explain how to use preconditioners for deriving tractable approximate distributions. In Sections 3.1 and 4, we then describe the underlying theory, which we call the generalized support theory for graphical models. In Section 5 we present experiments that compare the new approximation schemes to some of the standard variational and optimization based methods.

## 2 Notation and Background

Consider a graph $G = (V, E)$, where $V$ denotes the set of nodes and $E$ denotes the set of edges. Let $X_i$ be a random variable associated with node $i$, for $i \in V$, yielding a random vector $X = \{X_1, \ldots, X_n\}$. Let $\phi = \{\phi_\alpha, \alpha \in I\}$ denote the set of *potential functions* or *sufficient statistics*, for a set $I$ of cliques in G. Associated with $\phi$ is a vector of parameters $\theta = \{\theta_\alpha, \alpha \in I\}$. With this notation, the exponential family of distributions of $X$, associated with $\phi$ and $G$, is given by

$$p(x; \theta) = \exp \left( \sum_\alpha \theta_\alpha \phi_\alpha - \Psi(\theta) \right). \qquad (1)$$

For traditional reasons through connections with statistical physics, $Z = \exp \Psi(\theta)$ is called the *partition function*. As discussed in (Yedidia et al., 2001), at the expense in increasing the state space one can assume without loss of generality that the graphical model is a pairwise Markov random field, *i.e.*, the set of cliques $I$ is the set of edges $\{(s, t) \in E\}$. We shall assume a pairwise random field, and thus can express the potential function and parameter vectors in more compact form as matrices:

$$\Theta := \begin{pmatrix} \theta_{11} & \ldots & \theta_{1n} \\ \vdots & \vdots & \vdots \\ \theta_{n1} & \ldots & \theta_{nn} \end{pmatrix} \quad \Phi(x) := \begin{pmatrix} \phi_{11}(x_1, x_1) & \ldots & \phi_{1n}(x_1, x_n) \\ \vdots & \vdots & \vdots \\ \phi_{n1}(x_n, x_1) & \ldots & \phi_{nn}(x_n, x_n) \end{pmatrix} \qquad (2)$$

In the following we will denote the trace of the product of two matrices $A$ and $B$ by the inner product $\langle\langle A, B \rangle\rangle$. Assuming that each $X_i$ is finite-valued, the partition function $Z(\Theta)$ is then given by $Z(\Theta) = \sum_{x \in \chi} \exp \langle\langle \Theta, \Phi(x) \rangle\rangle$. The computation of $Z(\Theta)$ has a complexity exponential in the tree-width of the graph $G$ and hence is intractable for large graphs. Our goal is to obtain rigorous upper and lower bounds for this partition function, which can then be used to obtain rigorous upper and lower bounds for general event probabilities; this is discussed further in (Ravikumar and Lafferty, 2004).

### 2.1 Preconditioners in Linear Systems

Consider a linear system, $Ax = c$, where the variable $x$ is $n$ dimensional, and $A$ is an $n \times n$ matrix with $m$ non-zero entries. Solving for $x$ via direct methods such as Gaussian elimination has a computational complexity $O(n^3)$, which is impractical for large values of $n$. Multiplying both sides of the linear system by the inverse of an invertible matrix $B$, we get an equivalent "preconditioned" system, $B^{-1}Ax = B^{-1}c$. If $B$ is similar to $A$, $B^{-1}A$ is in turn similar to $I$, the identity matrix, making the preconditioned system easier to solve. Such an approximating matrix $B$ is called a preconditioner.

The computational complexity of preconditioned conjugate gradient is given by

$$T(A) = \sqrt{\kappa(A, B)} \, (m + T(B)) \log \left( \frac{1}{\epsilon} \right) \qquad (3)$$

where $T(A)$ is the time required for an $\epsilon$-approximate solution; $\kappa(A, B)$ is the *condition number* of $A$ and $B$ which intuitively corresponds to the quality of the approximation $B$, and $T(B)$ is the time required to solve $By = c$.

Recent developments in the theory of preconditioners are in part based on *support graph theory*, where the linear system matrix is viewed as the Laplacian of a graph, and graph-based techniques can be used to obtain good approximations. While these methods require diagonally dominant matrices ($A_{ii} \geq \sum_{j \neq i} |A_{ij}|$), they yield "ultra-sparse" (tree plus a constant number of edges) preconditioners with a low condition number. In our

experiments, we use two elementary tree-based preconditioners in this family, Vaidya's Spanning Tree preconditioner Vaidya (1990), and Gremban-Miller's Support Tree preconditioner Gremban (1996).

## 3 Graphical Model Preconditioners

Our proposed framework follows the generalized mean field intuition of looking at sparse graph approximations of the original graph, but solving a different optimization problem. We begin by outlining the basic idea, and then develop the underlying theory.

Consider the graphical model with graph $G$, potential-function matrix $\Phi(x)$, and parameter matrix $\Theta$. For purposes of intuition, think of the graphical model "energy" $\langle\langle \Theta, \Phi(x)\rangle\rangle$ as the matrix norm $x^\top \Theta x$. We would like to obtain a sparse approximation $B$ for $\Theta$. If $B$ approximates $\Theta$ well, then the condition number $\kappa$ is small:

$$\kappa(\Theta, B) \;=\; \max_x \frac{x^\top \Theta x}{x^\top B x} \;\Big/\; \min_x \frac{x^\top \Theta x}{x^\top B x} \;=\; \lambda_{max}(\Theta, B) \,/\lambda_{min}(\Theta, B) \qquad (4)$$

This suggests the following procedure for approximate inference. First, choose a matrix $B$ that minimizes the condition number with $\Theta$ (rather than KL divergence as in mean-field). Then, scale $B$ appropriately, as detailed in the following sections. Finally, use the scaled matrix $B$ as the parameter matrix for approximate inference. Note that if $B$ corresponds to a tree, approximate inference has linear time complexity.

### 3.1 Generalized Eigenvalue Bounds

Given a graphical model with graph $G$, potential-function matrix $\Phi(x)$, and parameter matrix $\Theta$, our goal is to obtain parameter matrices $\Theta_U$ and $\Theta_L$, corresponding to sparse graph approximations of $G$, such that

$$Z(\Theta_L) \;\leq\; Z(\Theta) \;\leq\; Z(\Theta_U). \qquad (5)$$

That is, the partition functions of the sparse graph parameter matrices $\Theta_U$ and $\Theta_L$ are upper and lower bounds, respectively, of the partition function of the original graph. However, we will instead focus on a seemingly much *stronger* condition; in particular, we will look for $\Theta_L$ and $\Theta_U$ that satisfy

$$\langle\langle \Theta_L, \Phi(x)\rangle\rangle \;\leq\; \langle\langle \Theta, \Phi(x)\rangle\rangle \;\leq\; \langle\langle \Theta_U, \Phi(x)\rangle\rangle \qquad (6)$$

for all $x$. By monotonicity of $\exp$, this stronger condition implies condition (5) on the partition function, by summing over the values of $X$. However, this stronger condition will give us greater flexibility, and rigorous bounds for general event probabilities since then

$$\frac{\exp\langle\langle \Theta_L, \Phi(x)\rangle\rangle}{Z(\Theta_U)} \;\leq\; p(x; \Theta) \;\leq\; \frac{\exp\langle\langle \Theta_U, \Phi(x)\rangle\rangle}{Z(\Theta_L)}. \qquad (7)$$

In contrast, while variational methods give bounds on the log partition function, the derived bounds on general event probabilities via the variational parameters are only heuristic.

Let $\mathcal{S}$ be a set of sparse graphs; for example, $\mathcal{S}$ may be the set of all trees. Focusing on the upper bound, we for now would like to obtain a graph $G' \in \mathcal{S}$ with parameter matrix $B$, which approximates $G$, and whose partition function upper bounds the partition function of the original graph. Following (6), we require,

$$\langle\langle \Theta, \Phi(x)\rangle\rangle \;\leq\; \langle\langle B, \Phi(x)\rangle\rangle, \text{ such that } G(B) \in \mathcal{S} \qquad (8)$$

where $G(B)$ denotes the graph corresponding to the parameter matrix $B$. Now, we would like the distribution corresponding to $B$ to be as close as possible to the distribution corresponding to $\Theta$; that is, $\langle\langle B, \Phi(x)\rangle\rangle$ should not only upper bound $\langle\langle \Theta, \Phi(x)\rangle\rangle$ but should be

close to it. The distance measure we use for this is the minimax distance. In other words, while the upper bound requires that

$$\frac{\langle\!\langle \Theta, \Phi(x) \rangle\!\rangle}{\langle\!\langle B, \Phi(x) \rangle\!\rangle} \leq 1, \tag{9}$$

we would like

$$\min_x \frac{\langle\!\langle \Theta, \Phi(x) \rangle\!\rangle}{\langle\!\langle B, \Phi(x) \rangle\!\rangle} \tag{10}$$

to be as high as possible. Expressing these desiderata in the form of an optimization problem, we have

$$B^\star = \arg\max_{B:\, G(B)\in\mathcal{S}} \min_x \frac{\langle\!\langle \Theta, \Phi(x) \rangle\!\rangle}{\langle\!\langle B, \Phi(x) \rangle\!\rangle}, \quad \text{such that } \frac{\langle\!\langle \Theta, \Phi(x) \rangle\!\rangle}{\langle\!\langle B, \Phi(x) \rangle\!\rangle} \leq 1.$$

Before solving this problem, we first make some definitions, which are generalized versions of standard concepts in linear systems theory.

**Definition 3.1.** *For a pairwise Markov random field with potential function matrix $\Phi(x)$; the generalized eigenvalues of a pair of parameter matrices $(A, B)$ are defined as*

$$\lambda_{max}^{\Phi}(A, B) = \max_{x:\, \langle\!\langle B,\Phi(x)\rangle\!\rangle \neq 0} \frac{\langle\!\langle A, \Phi(x) \rangle\!\rangle}{\langle\!\langle B, \Phi(x) \rangle\!\rangle} \tag{11}$$

$$\lambda_{min}^{\Phi}(A, B) = \min_{x:\, \langle\!\langle B,\Phi(x)\rangle\!\rangle \neq 0} \frac{\langle\!\langle A, \Phi(x) \rangle\!\rangle}{\langle\!\langle B, \Phi(x) \rangle\!\rangle}. \tag{12}$$

Note that

$$\lambda_{max}^{\Phi}(A, \alpha B) = \max_{x:\, \langle\!\langle \alpha B,\Phi(x)\rangle\!\rangle \neq 0} \frac{\langle\!\langle A, \Phi(x) \rangle\!\rangle}{\langle\!\langle \alpha B, \Phi(x) \rangle\!\rangle} \tag{13}$$

$$= \frac{1}{\alpha} \max_{x:\, \langle\!\langle B,\Phi(x)\rangle\!\rangle \neq 0} \frac{\langle\!\langle A, \Phi(x) \rangle\!\rangle}{\langle\!\langle B, \Phi(x) \rangle\!\rangle} = \alpha^{-1}\lambda_{max}^{\Phi}(A, B). \tag{14}$$

We state the basic properties of the generalized eigenvalues in the following lemma.

**Lemma 3.2.** *The generalized eigenvalues satisfy*

$$\lambda_{min}^{\Phi}(A, B) \leq \frac{\langle\!\langle A, \Phi(x) \rangle\!\rangle}{\langle\!\langle B, \Phi(x) \rangle\!\rangle} \leq \lambda_{max}^{\Phi}(A, B) \tag{15}$$

$$\lambda_{max}^{\Phi}(A, \alpha B) = \alpha^{-1}\lambda_{max}^{\Phi}(A, B) \tag{16}$$

$$\lambda_{min}^{\Phi}(A, \alpha B) = \alpha^{-1}\lambda_{min}^{\Phi}(A, B) \tag{17}$$

$$\lambda_{min}^{\Phi}(A, B) = \frac{1}{\lambda_{max}^{\Phi}(B, A)}. \tag{18}$$

In the following, we will use $A$ to generically denote the parameter matrix $\Theta$ of the model. We can now rewrite the optimization problem for the upper bound in equation (11) as

$$(\text{Problem } \Lambda_1) \qquad \max_{B:\, G(B)\in\mathcal{S}} \lambda_{min}^{\Phi}(A, B), \quad \text{such that } \lambda_{max}^{\Phi}(A, B) \leq 1 \tag{19}$$

We shall express the optimal solution of Problem $\Lambda_1$ in terms of the optimal solution of a companion problem. Towards that end, consider the optimization problem

$$(\text{Problem } \Lambda_2) \qquad \min_{C:\, G(C)\in\mathcal{S}} \frac{\lambda_{max}^{\Phi}(A, C)}{\lambda_{min}^{\Phi}(A, C)}. \tag{20}$$

The following proposition shows the sense in which these problems are equivalent.

**Proposition 3.3.** *If $\widehat{C}$ attains the optimum in Problem $\Lambda_2$, then $\widetilde{C} = \lambda_{max}^{\Phi}(A, \widehat{C})\, \widehat{C}$ attains the optimum of Problem $\Lambda_1$.*

*Proof.* For any feasible solution $B$ of Problem $\Lambda_1$, we have

$$\lambda_{\min}^{\Phi}(A, B) \quad \leq \quad \frac{\lambda_{\min}^{\Phi}(A, B)}{\lambda_{\max}^{\Phi}(A, B)} \quad \text{(since } \lambda_{\max}^{\Phi}(A, B) \leq 1) \tag{21}$$

$$\leq \quad \frac{\lambda_{\min}^{\Phi}(A, \widehat{C})}{\lambda_{\max}^{\Phi}(A, \widehat{C})} \quad \text{(since } \widehat{C} \text{ is the optimum of Problem } \Lambda_2) \tag{22}$$

$$= \quad \lambda_{\min}^{\Phi}\left(A, \lambda_{\max}^{\Phi}(A, \widehat{C})\widehat{C}\right) \quad \text{(from Lemma 3.2)} \tag{23}$$

$$= \quad \lambda_{\min}^{\Phi}(A, \widetilde{C}). \tag{24}$$

Thus, $\widetilde{C}$ upper bounds all feasible solutions in Problem $\Lambda_1$. However, it itself is a feasible solution, since

$$\lambda_{\max}^{\Phi}(A, \widetilde{C}) \; = \; \lambda_{\max}^{\Phi}\left(A, \lambda_{\max}^{\Phi}(A, \widehat{C})\widehat{C}\right) \; = \; \frac{1}{\lambda_{\max}^{\Phi}(A, \widehat{C})}\lambda_{\max}^{\Phi}(A, \widehat{C}) \; = \; 1 \tag{25}$$

from Lemma 3.2. Thus, $\widetilde{C}$ attains the maximum in the upper bound Problem $\Lambda_1$.  $\square$

The analysis for obtaining an upper bound parameter matrix $B$ for a given parameter matrix $A$ carries over for the lower bound; we need to replace a maximin problem with a minimax problem. For the lower bound, we want a matrix $B$ such that

$$B_\star \;\; = \;\; \min_{B:\, G(B)\in\mathcal{S}} \;\; \max_{\{x:\, \langle\langle B, \Phi(x)\rangle\rangle \neq 0\}} \frac{\langle\langle A, \Phi(x)\rangle\rangle}{\langle\langle B, \Phi(x)\rangle\rangle}, \;\; \text{such that} \;\; \frac{\langle\langle A, \Phi(x)\rangle\rangle}{\langle\langle B, \Phi(x)\rangle\rangle} \geq 1 \tag{26}$$

This leads to the following lower bound optimization problem.

$$\text{(Problem } \Lambda_3) \qquad \min_{B:\, G(B)\in\mathcal{S}} \lambda_{\max}^{\Phi}(A, B), \;\; \text{such that} \;\; \lambda_{\min}^{\Phi}(A, B) \geq 1. \tag{27}$$

The proof of the following statement closely parallels the proof of Proposition 3.3.

**Proposition 3.4.** *If $\hat{C}$ attains the optimum in Problem $\Lambda_2$, then $\underline{C} = \lambda_{min}^{\Phi}(A, \hat{C})\hat{C}$ attains the optimum of the lower bound Problem $\Lambda_3$.*

Finally, we state the following basic lemma, whose proof is easily verified.

**Lemma 3.5.** *For any pair of parameter-matrices $(A, B)$, we have*

$$\left\langle\!\!\left\langle \lambda_{min}^{\Phi}(A, B)B, \Phi(x)\right\rangle\!\!\right\rangle \; \leq \; \left\langle\!\!\left\langle A, \Phi(x)\right\rangle\!\!\right\rangle \; \leq \; \left\langle\!\!\left\langle \lambda_{max}^{\Phi}(A, B)B, \Phi(x)\right\rangle\!\!\right\rangle. \tag{28}$$

## 3.2 Main Procedure

We now have in place the machinery necessary to describe the procedure for solving the main problem in equation (6), to obtain upper and lower bound matrices for a graphical model. Lemma 3.5 shows how to obtain upper and lower bound parameter matrices with respect to any matrix $B$, given a parameter matrix $A$, by solving a generalized eigenvalue problem. Propositions 3.3 and 3.4 tell us, in principle, how to obtain the optimal such upper and lower bound matrices. We thus have the following procedure. First, obtain a parameter matrix $C$ such that $G(C) \in \mathcal{S}$, which minimizes $\lambda_{\max}^{\Phi}(\Theta, C)/\lambda_{\min}^{\Phi}(\Theta, C)$. Then $\lambda_{\max}^{\Phi}(\Theta, C)\, C$ gives the optimal upper bound parameter matrix and $\lambda_{\min}^{\Phi}(\Theta, C)\, C$ gives the optimal lower bound parameter matrix. However, as things stand, this recipe appears to be even more challenging to work with than the generalized mean field procedures. The difficulty lies in obtaining the matrix $C$. In the following section we offer a series of relaxations that help to simplify this task.

# 4   Generalized Support Theory for Graphical Models

In what follows, we begin by assuming that the potential function matrix is positive semi-definite, $\Phi(x) \succeq 0$, and later extend our results to general $\Phi$.

**Definition 4.1.** *For a pairwise MRF with potential function matrix $\Phi(x) \succeq 0$, the generalized support number of a pair of parameter matrices $(A, B)$, where $B \succeq 0$, is*

$$\sigma^\Phi(A, B) = \min \left\{ \tau \in \mathbf{R} \,\middle|\, \langle\!\langle \tau B, \Phi(x) \rangle\!\rangle \geq \langle\!\langle A, \Phi(x) \rangle\!\rangle \text{ for all } x \right\} \tag{29}$$

The generalized support number can be thought of as the "number of copies" $\tau$ of $B$ required to "support" $A$ so that $\langle\!\langle \tau B - A, \Phi(x) \rangle\!\rangle \geq 0$. The usefulness of this definition is demonstrated by the following result.

**Proposition 4.2.** *If $B \succeq 0$ then $\lambda^\Phi_{max}(A, B) \leq \sigma^\Phi(A, B)$.*

*Proof.* From the definition of the generalized support number for a graphical model, we have that $\langle\!\langle \sigma^\Phi(A, B)B - A, \Phi(x) \rangle\!\rangle \geq 0$. Now, since we assume that $\Phi(x) \succeq 0$, if also $B \succeq 0$ then $\langle\!\langle B, \Phi(x) \rangle\!\rangle \geq 0$. Therefore, it follows that $\frac{\langle\!\langle A, \Phi(x) \rangle\!\rangle}{\langle\!\langle B, \Phi(x) \rangle\!\rangle} \leq \sigma^\Phi(A, B)$, and thus

$$\lambda^\Phi_{max}(A, B) = \max_x \frac{\langle\!\langle A, \Phi(x) \rangle\!\rangle}{\langle\!\langle B, \Phi(x) \rangle\!\rangle} \leq \sigma^\Phi(A, B) \tag{30}$$

giving the statement of the proposition.  □

This leads to our first relaxation of the generalized eigenvalue bound for a model. From Lemma 3.2 and Proposition 4.2 we see that

$$\frac{\lambda^\Phi_{max}(A, B)}{\lambda^\Phi_{min}(A, B)} = \lambda^\Phi_{max}(A, B)\lambda^\Phi_{max}(B, A) \leq \sigma^\Phi(A, B)\sigma^\Phi(B, A) \tag{31}$$

Thus, this result suggests that to approximate the graphical model $(\Theta, \Phi)$ we can search for a parameter matrix $B^\star$, with corresponding simple graph $G(B^\star) \in \mathcal{S}$, such that

$$B^\star = \arg\min_B \sigma^\Phi(\Theta, B)\sigma^\Phi(B, \Theta) \tag{32}$$

While this relaxation may lead to effective bounds, we will now go further, to derive an additional relaxation that relates our generalized graphical model support number to the "classical" support number.

**Proposition 4.3.** *For a potential function matrix $\Phi(x) \succeq 0$, $\sigma^\Phi(A, B) \leq \sigma(A, B)$, where $\sigma(A, B) = \min\{\tau \,|\, (\tau B - A) \succeq 0\}$.*

*Proof.* Since $\sigma(A, B)B - A \succeq 0$ by definition and $\Phi(x) \succeq 0$ by assumption, we have that $\langle\!\langle \sigma(A, B)B - A, \Phi(x) \rangle\!\rangle \geq 0$. Therefore, $\sigma^\Phi(A, B) \leq \sigma(A, B)$ from the definition of generalized support number.  □

The above result reduces the problem of approximating a graphical model to the problem of minimizing classical support numbers, the latter problem being well-studied in the scientific computing literature (Boman and Hendrickson, 2003; Bern et al., 2001), where the expression $\sigma(A, C)\sigma(C, A)$ is called the *condition number*, and a matrix that minimizes it within a simple family of graphs is called a *preconditioner*. We can thus plug in any algorithm for finding a sparse preconditioner for $\Theta$, carrying out the optimization

$$B^\star = \arg\min_B \sigma(\Theta, B)\,\sigma(B, \Theta) \tag{33}$$

and then use that matrix $B^\star$ in our basic procedure.

One example is Vaidya's preconditioner Vaidya (1990), which is essentially the maximum spanning tree of the graph. Another is the support tree of Gremban (1996), which introduces Steiner nodes, in this case auxiliary nodes introduced via a recursive partitioning of the graph. We present experiments with these basic preconditioners in the following section.

Before turning to the experiments, we comment that our generalized support number analysis assumed that the potential function matrix $\Phi(x)$ was positive semi-definite. The case when it is not can be handled as follows. We first add a large positive diagonal matrix $D$ so that $\Phi'(x) = \Phi(x) + D \succeq 0$. Then, for a given parameter matrix $\Theta$, we use the above machinery to get an upper bound parameter matrix $B$ such that

$$\langle\langle A, \Phi(x) + D \rangle\rangle \le \langle\langle B, \Phi(x) + D \rangle\rangle \;\Rightarrow\; \langle\langle A, \Phi(x) \rangle\rangle \le \langle\langle B, \Phi(x) \rangle\rangle + \langle\langle B - A, D \rangle\rangle. \tag{34}$$

Exponentiating and summing both sides over x, we then get the required upper bound for the parameter matrix A; the same can be done for the lower bound.

## 5 Experiments

As the previous sections detailed, the preconditioner based bounds are in principle quite easy to compute—we compute a sparse preconditioner for the parameter matrix (typically $O(n)$ to $O(n^3)$) and use the preconditioner as the parameter matrix for the bound computation (which is linear if the preconditioner matrix corresponds to a tree). This yields a simple, non-iterative deterministic procedure as compared to the more complex propagation-based or iterative update procedures. In this section we evaluate these bounds on small graphical models for which exact answers can be readily computed, and compare the bounds to variational approximations.

We show simulation results averaged over a randomly generated set of graphical models. The graphs used were 2D grid graphs, and the edge potentials were selected according to a uniform distribution Uniform$(-2d_{coup}, 0)$ for various coupling strengths $d_{coup}$. We report the relative error, (bound − log-partition-function)/log-partition-function.

As a baseline, we use the mean field and structured mean field methods for the lower bound, and the Wainwright et al. (2003) tree-reweighted belief propagation approximation for the upper bound. For the preconditioner based bounds, we use two very simple preconditioners, (a) Vaidya's maximum spanning tree preconditioner (Vaidya, 1990), which assumes the input parameter matrix to be a Laplacian, and (b) Gremban (1996)'s support tree preconditioner, which also gives a sparse parameter matrix corresponding to a tree, with Steiner (auxiliary) nodes. To compute bounds over these larger graphs with Steiner nodes we average an internal node over its children; this is the technique used with such preconditioners for solving linear systems. We note that these preconditioners are quite basic, and the use of better preconditioners (yielding a better condition number) has the potential to achieve much better bounds, as shown in Propositions 3.3 and 3.4. We also reiterate that while our approach can be used to derive bounds on event probabilities, the variational methods yield bounds only for the partition function, and only apply heuristically to estimating simple event probabilities such as marginals.

As the plots in Figure 1 show, even for the simple preconditioners used, the new bounds are quite close to the actual values, outperforming the mean field method and giving comparable results to the tree-reweighted belief propagation method. The spanning tree preconditioner provides a good lower bound, while the support tree preconditioner provides a good upper bound, however not as tight as the bound obtained using tree-reweighted belief propagation. Although we cannot compute the exact solution for large graphs, we can

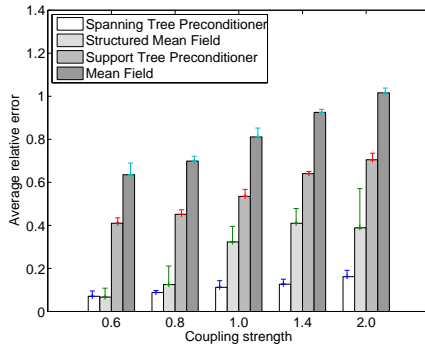

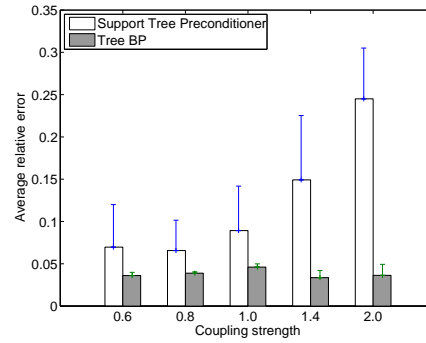

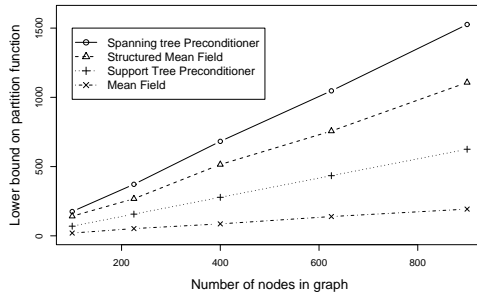

Figure 1: Comparison of lower bounds (top left), and upper bounds (top right) for small grid graphs, and lower bounds for grid graphs of increasing size (left).

compare bounds. The bottom plot of Figure 1 compares lower bounds for graphs with up to 900 nodes; a larger bound is necessarily tighter, and the preconditioner bounds are seen to outperform mean field.

## Acknowledgments

We thank Gary Miller for helpful discussions. Research supported in part by NSF grants IIS-0312814 and IIS-0427206.

## References

M. Bern, J. R. Gilbert, B. Hendrickson, N. Nguyen, and S. Toledo. Support-graph preconditioners. Submitted to *SIAM J. Matrix Anal. Appl.*, 2001.

E. G. Boman and B. Hendrickson. Support theory for preconditioning. *SIAM Journal on Matrix Analysis and Applications*, 25, 2003.

K. Gremban. Combinatorial preconditioners for sparse, symmetric, diagonally dominant linear systems. *Ph.D. Thesis, Carnegie Mellon University, 1996*, 1996.

P. Ravikumar and J. Lafferty. Variational Chernoff bounds for graphical models. *Proceedings of Uncertainty in Artificial Intelligence (UAI)*, 2004.

P. M. Vaidya. Solving linear equations with symmetric diagonally dominant matrices by constructing good preconditioners. 1990. Unpublished manuscript, UIUC.

M. J. Wainwright, T. Jaakkola, and A. S. Willsky. Tree-reweighted belief propagation and approximate ML estimation by pseudo-moment matching. *9th Workshop on Artificial Intelligence and Statistics*, 2003.

J. S. Yedidia, W. T. Freeman, and Y. Weiss. Understanding belief propagation and its generalizations. *IJCAI 2001 Distinguished Lecture track*, 2001.